# Hidden Markov decision trees

**Michael I. Jordan\***, **Zoubin Ghahramani†**, and **Lawrence K. Saul\***
{jordan,zoubin,lksaul}@psyche.mit.edu

\*Center for Biological and Computational Learning
Massachusetts Institute of Technology
Cambridge, MA USA 02139

†Department of Computer Science
University of Toronto
Toronto, ON Canada M5S 1A4

## Abstract

We study a time series model that can be viewed as a decision tree with Markov temporal structure. The model is intractable for exact calculations, thus we utilize variational approximations. We consider three different distributions for the approximation: one in which the Markov calculations are performed exactly and the layers of the decision tree are decoupled, one in which the decision tree calculations are performed exactly and the time steps of the Markov chain are decoupled, and one in which a Viterbi-like assumption is made to pick out a single most likely state sequence. We present simulation results for artificial data and the Bach chorales.

## 1 Introduction

Decision trees are regression or classification models that are based on a nested decomposition of the input space. An input vector $\mathbf{x}$ is classified recursively by a set of "decisions" at the nonterminal nodes of a tree, resulting in the choice of a terminal node at which an output $\mathbf{y}$ is generated. A statistical approach to decision tree modeling was presented by Jordan and Jacobs (1994), where the decisions were treated as hidden multinomial random variables and a likelihood was computed by summing over these hidden variables. This approach, as well as earlier statistical analyses of decision trees, was restricted to independently, identically distributed data. The goal of the current paper is to remove this restriction; we describe a generalization of the decision tree statistical model which is appropriate for time series.

The basic idea is straightforward—we assume that each decision in the decision tree is dependent on the decision taken at that node at the previous time step. Thus we augment the decision tree model to include Markovian dynamics for the decisions.

For simplicity we restrict ourselves to the case in which the decision variable at a given nonterminal is dependent only on the same decision variable at the same nonterminal at the previous time step. It is of interest, however, to consider more complex models in which inter-nonterminal pathways allow for the possibility of various kinds of synchronization.

Why should the decision tree model provide a useful starting point for time series modeling? The key feature of decision trees is the nested decomposition. If we view each nonterminal node as a basis function, with support given by the subset of possible input vectors $\mathbf{x}$ that arrive at the node, then the support of each node is the union of the support associated with its children. This is reminiscent of wavelets, although without the strict condition of multiplicative scaling. Moreover, the regions associated with the decision tree are polygons, which would seem to provide a useful generalization of wavelet-like decompositions in the case of a high-dimensional input space.

The architecture that we describe in the current paper is fully probabilistic. We view the decisions in the decision tree as multinomial random variables, and we are concerned with calculating the posterior probabilities of the time sequence of hidden decisions given a time sequence of input and output vectors. Although such calculations are tractable for decision trees and for hidden Markov models separately, the calculation is intractable for our model. Thus we must make use of approximations. We utilize the partially factorized variational approximations described by Saul and Jordan (1996), which allow tractable substructures (e.g., the decision tree and Markov chain substructures) to be handled via exact methods, within an overall approximation that guarantees a lower bound on the log likelihood.

## 2   Architectures

### 2.1   Probabilistic decision trees

The "hierarchical mixture of experts" (HME) model (Jordan & Jacobs, 1994) is a decision tree in which the decisions are modeled probabilistically, as are the outputs. The total probability of an output given an input is the sum over all paths in the tree from the input to the output. The HME model is shown in the graphical model formalism in Figure 2.1. Here a node represents a random variable, and the links represent probabilistic dependencies. A conditional probability distribution is associated with each node in the graph, where the conditioning variables are the node's parents.

Let $\mathbf{z}^1$, $\mathbf{z}^2$, and $\mathbf{z}^3$ denote the (multinomial) random variables corresponding to the first, second and third levels of the decision tree.[1] We associate multinomial probabilities $P(\mathbf{z}^1|\mathbf{x}, \eta^1)$, $P(\mathbf{z}^2|\mathbf{x}, \mathbf{z}^1, \eta^2)$, and $P(\mathbf{z}^3|\mathbf{x}, \mathbf{z}^1, \mathbf{z}^2, \eta^3)$ with the decision nodes, where $\eta^1, \eta^2$, and $\eta^3$ are parameters (e.g., Jordan and Jacobs utilized soft-max transformations of linear functions of $\mathbf{x}$ for these probabilities). The leaf probabilities $P(\mathbf{y}|\mathbf{x}, \mathbf{z}^1, \mathbf{z}^2, \mathbf{z}^3, \theta)$ are arbitrary conditional probability models; e.g., linear/Gaussian models for regression problems.

The key calculation in the fitting of the HME model to data is the calculation of the posterior probabilities of the hidden decisions given the clamped values of $\mathbf{x}$ and $\mathbf{y}$. This calculation is a recursion extending upward and downward in the tree, in which the posterior probability at a given nonterminal is the sum of posterior probabilities associated with its children. The recursion can be viewed as a special

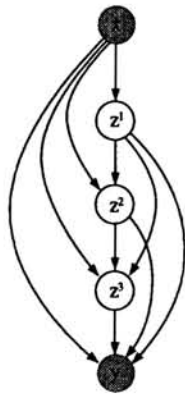

Figure 1: The hierarchical mixture of experts as a graphical model. The E step of the learning algorithm for HME's involves calculating the posterior probabilities of the hidden (unshaded) variables given the observed (shaded) variables.

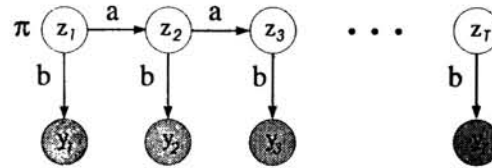

Figure 2: An HMM as a graphical model. The transition matrix appears on the horizontal links and the output probability distribution on the vertical links. The E step of the learning algorithm for HMM's involves calculating the posterior probabilities of the hidden (unshaded) variables given the observed (shaded) variables.

case of generic algorithms for calculating posterior probabilities on directed graphs (see, e.g., Shachter, 1990).

## 2.2 Hidden Markov models

In the graphical model formalism a hidden Markov model (HMM; Rabiner, 1989) is represented as a chain structure as shown in Figure 2.1. Each state node is a multinomial random variable $z_t$. The links between the state nodes are parameterized by the transition matrix $a(z_t|z_{t-1})$, assumed homogeneous in time. The links between the state nodes $z_t$ and output nodes $y_t$ are parameterized by the output probability distribution $b(y_t|z_t)$, which in the current paper we assume to be Gaussian with (tied) covariance matrix $\Sigma$.

As in the HME model, the key calculation in the fitting of the HMM to observed data is the calculation of the posterior probabilities of the hidden state nodes given the sequence of output vectors. This calculation—the E step of the Baum-Welch algorithm—is a recursion which proceeds forward or backward in the chain.

## 2.3 Hidden Markov decision trees

We now marry the HME and the HMM to produce the hidden Markov decision tree (HMDT) shown in Figure 3. This architecture can be viewed in one of two ways: (a) as a time sequence of decision trees in which the decisions in a given decision tree depend probabilistically on the decisions in the decision tree at the preceding moment in time; (b) as an HMM in which the state variable at each moment in time is factorized (cf. Ghahramani & Jordan, 1996) and the factors are coupled vertically to form a decision tree structure.

Let the state of the Markov process defining the HMDT be given by the values of hidden multinomial decisions $z_t^1$, $z_t^2$, and $z_t^3$, where the superscripts denote the level of the decision tree (the vertical dimension) and the subscripts denote the time (the horizontal dimension). Given our assumption that the state transition matrix has only intra-level Markovian dependencies, we obtain the following expression for the

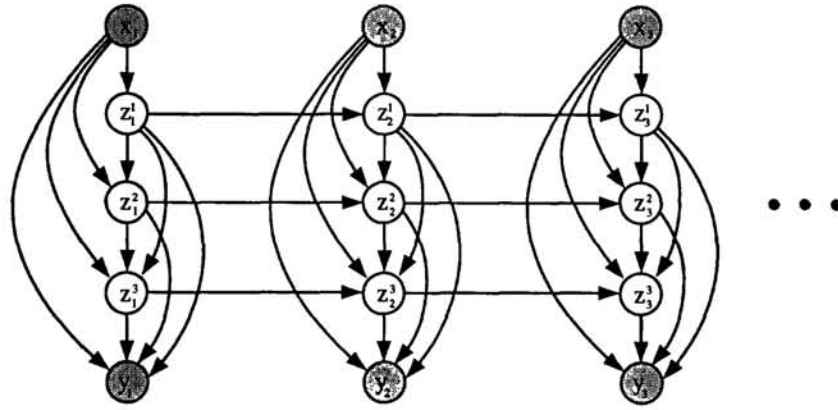

Figure 3: The HMDT model is an HME decision tree (in the vertical dimension) with Markov time dependencies (in the horizontal dimension).

HMDT probability model:

$$P(\{z_t^1, z_t^2, z_t^3\}, \{y_t\}|\{x_t\}) = \pi^1(z_1^1|x_1)\pi^2(z_1^2|x_1, z_1^1)\pi^3(z_1^3|x_1, z_1^1, z_1^2)$$

$$\prod_{t=2}^{T} a^1(z_t^1|x_t, z_{t-1}^1)a^2(z_t^2|x_t, z_{t-1}^2, z_t^1)a^3(z_t^3|x_t, z_{t-1}^3, z_t^1, z_t^2) \prod_{t=1}^{T} b(y_t|x_t, z_t^1, z_t^2, z_t^3)$$

Summing this probability over the hidden values $z_t^1, z_t^2$, and $z_t^3$ yields the HMDT likelihood.

The HMDT is a 2-D lattice with inhomogeneous field terms (the observed data). It is well-known that such lattice structures are intractable for exact probabilistic calculations. Thus, although it is straightforward to write down the EM algorithm for the HMDT and to write recursions for the calculations of posterior probabilities in the E step, these calculations are likely to be too time-consuming for practical use (for $T$ time steps, $K$ values per node and $M$ levels, the algorithm scales as $O(K^{M+1}T)$). Thus we turn to methods that allow us to approximate the posterior probabilities of interest.

## 3 Algorithms

### 3.1 Partially factorized variational approximations

Completely factorized approximations to probability distributions on graphs can often be obtained variationally as mean field theories in physics (Parisi, 1988). For the HMDT in Figure 3, the completely factorized mean field approximation would delink all of the nodes, replacing the interactions with constant fields acting at each of the nodes. This approximation, although useful, neglects to take into account the existence of efficient algorithms for tractable substructures in the graph.

Saul and Jordan (1996) proposed a refined mean field approximation to allow interactions associated with tractable substructures to be taken into account. The basic idea is to associate with the intractable distribution $P$ a simplified distribution $Q$ that retains certain of the terms in $P$ and neglects others, replacing them with parameters $\mu_i$ that we will refer to as "variational parameters." Graphically the method can be viewed as deleting arcs from the original graph until a forest of tractable substructures is obtained. Arcs that remain in the simplified graph

correspond to terms that are retained in $Q$; arcs that are deleted correspond to variational parameters.

To obtain the best possible approximation of $P$ we minimize the Kullback-Liebler divergence $KL(Q\|P)$ with respect to the parameters $\mu_i$. The result is a coupled set of equations that are solved iteratively. These equations make reference to the values of expectations of nodes in the tractable substructures; thus the (efficient) algorithms that provide such expectations are run as subroutines. Based on the posterior expectations computed under $Q$, the parameters defining $P$ are adjusted. The algorithm as a whole is guaranteed to increase a lower bound on the log likelihood.

### 3.2 A forest of chains

The HMDT can be viewed as a coupled set of chains, with couplings induced directly via the decision tree structure and indirectly via the common coupling to the output vector. If these couplings are removed in the variational approximation, we obtain a $Q$ distribution whose graph is a forest of chains. There are several ways to parameterize this graph; in the current paper we investigate a parameterization with time-varying transition matrices and time-varying fields. Thus the $Q$ distribution is given by

$$Q(\{z_t^1, z_t^2, z_t^3\} \mid \{y_t\}, \{x_t\}) = \frac{1}{Z_Q} \prod_{t=2}^{T} \tilde{a}_t^1(z_t^1|z_{t-1}^1)\tilde{a}_t^2(z_t^2|z_{t-1}^2)\tilde{a}_t^3(z_t^3|z_{t-1}^3)$$
$$\prod_{t=1}^{T} \tilde{q}_t^1(z_t^1)\tilde{q}_t^2(z_t^2)\tilde{q}_t^3(z_t^3)$$

where $\tilde{a}_t^i(z_t^i|z_{t-1}^i)$ and $\tilde{q}_t^i(z_t^i)$ are potentials that provide the variational parameterization.

### 3.3 A forest of decision trees

Alternatively we can drop the horizontal couplings in the HMDT and obtain a variational approximation in which the decision tree structure is handled exactly and the Markov structure is approximated. The $Q$ distribution in this case is

$$Q(\{z_t^1, z_t^2, z_t^3\} \mid \{y_t\}, \{x_t\}) = \prod_{t=1}^{T} \tilde{r}_t^1(z_1^1)\tilde{r}_t^2(z_1^2|z_1^1)\tilde{r}_t^3(z_1^3|z_1^1, z_1^2)$$

Note that a decision tree is a fully coupled graphical model; thus we can view the partially factorized approximation in this case as a completely factorized mean field approximation on "super-nodes" whose configurations include all possible configurations of the decision tree.

### 3.4 A Viterbi-like approximation

In hidden Markov modeling it is often found that a particular sequence of states has significantly higher probability than any other sequence. In such cases the Viterbi algorithm, which calculates only the most probable path, provides a useful computational alternative.

We can develop a Viterbi-like algorithm by utilizing an approximation $Q$ that assigns probability one to a single path $\{\bar{z}_t^1, \bar{z}_t^2, \bar{z}_t^3\}$:

$$Q(\{z_t^1, z_t^2, z_t^3\} \mid \{y_t\}, \{x_t\}) = \begin{cases} 1 & \text{if } z_t^i = \bar{z}_t^i, \ \forall t, i \\ 0 & \text{otherwise} \end{cases} \tag{1}$$

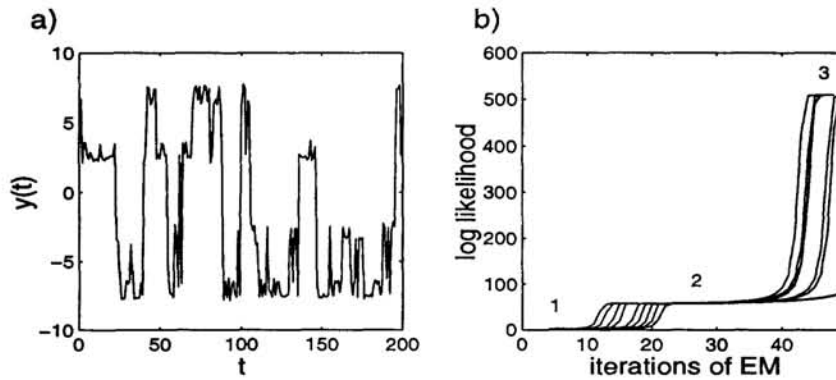

Figure 4: a) Artificial time series data. b) Learning curves for the HMDT.

Note that the entropy $Q \ln Q$ is zero, moreover the evaluation of the energy $Q \ln P$ reduces to substituting $\bar{z}_t^i$ for $z_t^i$ in $P$. Thus the variational approximation is particularly simple in this case. The resulting algorithm involves a subroutine in which a standard Viterbi algorithm is run on a single chain, with the other (fixed) chains providing field terms.

## 4  Results

We illustrate the HMDT on (1) an artificial time series generated to exhibit spatial and temporal structure at multiple scales, and (2) a domain which is likely to exhibit such structure naturally—the melody lines from J.S. Bach's chorales.

The artificial data was generated from a three level binary HMDT with no inputs, in which the root node determined coarse-scale shifts ($\pm 5$) in the time series, the middle node determined medium-scale shifts ($\pm 2$), and the bottom node determined fine-scale shifts ($\pm 0.5$) (Figure 4a). The temporal scales at these three nodes—as measured by the rate of convergence (second eigenvalue) of the transition matrices, with 0 (1) signifying immediate (no) convergence—were 0.85, 0.5, and 0.3, respectively.

We implemented forest-of-chains, forest-of-trees and Viterbi-like approximations. The learning curves for ten runs of the forest-of-chains approximation are shown in Figure 4b. Three plateau regions are apparent, corresponding to having extracted the coarse, medium, and fine scale structures of the time series. Five runs captured all three spatio-temporal scales at their correct level in the hierarchy; three runs captured the scales but placed them at incorrect nodes in the decision tree; and two captured only the coarse-scale structure.[2] Similar results were obtained with the Viterbi-like approximation. We found that the forest-of-trees approximation was not sufficiently accurate for these data.

The Bach chorales dataset consists of 30 melody lines with 40 events each.[3] Each discrete event encoded 6 attributes—start time of the event (st), pitch (pitch), duration (dur), key signature (key), time signature (time), and whether the event was under a fermata (ferm).

The chorales dataset was modeled with 3-level HMDTs with branching factors ($K$)

| | | Percent variance explained | | | | | Temporal scale | | |
|---|---|---|---|---|---|---|---|---|---|
| K | st | pitch | dur | key | time | ferm | level 1 | level 2 | level 3 |
| 2 | 3 | 6 | 6 | 84 | 95 | 0 | 1.00 | 1.00 | 0.51 |
| 3 | 22 | 38 | 7 | 93 | 99 | 0 | 1.00 | 0.96 | 0.85 |
| 4 | 55 | 48 | 36 | 96 | 99 | 5 | 1.00 | 1.00 | 0.69 |
| 5 | 57 | 41 | 41 | 97 | 99 | 61 | 1.00 | 0.95 | 0.75 |
| 6 | 70 | 40 | 58 | 94 | 99 | 10 | 1.00 | 0.93 | 0.76 |

Table 1: Hidden Markov decision tree models of the Bach chorales dataset: mean percentage of variance explained for each attribute and mean temporal scales at the different nodes.

2, 3, 4, 5, and 6 (3 runs at each size, summarized in Table 1). Thirteen out of 15 runs resulted in a coarse-to-fine progression of temporal scales from root to leaves of the tree. A typical run at branching factor 4, for example, dedicated the top level node to modeling the time and key signatures—attributes that are constant throughout any single chorale—the middle node to modeling start times, and the bottom node to modeling pitch or duration.

## 5    Conclusions

Viewed in the context of the burgeoning literature on adaptive graphical probabilistic models—which includes HMM's, HME's, CVQ's, IOHMM's (Bengio & Frasconi, 1995), and factorial HMM's—the HMDT would appear to be a natural next step. The HMDT includes as special cases all of these architectures, moreover it arguably combines their best features: factorized state spaces, conditional densities, representation at multiple levels of resolution and recursive estimation algorithms. Our work on the HMDT is in its early stages, but the earlier literature provides a reasonably secure foundation for its development.

## Footnotes

[1]Throughout the paper we restrict ourselves to three levels for simplicity of presentation.

[2]Note that it is possible to bias the ordering of the time scales by ordering the initial random values for the nodes of the tree; we did not utilize such a bias in this simulation.

[3]This dataset was obtained from the UCI Repository of Machine Learning Datasets.

## References

Bengio, Y., & Frasconi, P. (1995). An input output HMM architecture. In G. Tesauro, D. S. Touretzky & T. K. Leen, (Eds.), *Advances in Neural Information Processing Systems 7*, MIT Press, Cambridge MA.

Ghahramani, Z., & Jordan, M. I. (1996). Factorial hidden Markov models. In D. S. Touretzky, M. C. Mozer, & M. E. Hasselmo (Eds.), *Advances in Neural Information Processing Systems 8*, MIT Press, Cambridge MA.

Jordan, M. I., & Jacobs, R. A. (1994). Hierarchical mixtures of experts and the EM algorithm. *Neural Computation, 6*, 181–214.

Parisi, G. (1988). *Statistical Field Theory.* Redwood City, CA: Addison-Wesley.

Rabiner, L. (1989). A tutorial on hidden Markov models and selected application s in speech recognition. *Proceedings of the IEEE, 77*, 257–285.

Saul, L. K., & Jordan, M. I. (1996). Exploiting tractable substructures in intractable networks. In D. S. Touretzky, M. C. Mozer, & M. E. Hasselmo (Eds.), *Advances in Neural Information Processing Systems 8*, MIT Press, Cambridge MA.

Shachter, R. (1990). An ordered examination of influence diagrams. *Networks, 20*, 535–563.